# Permutation Complexity Bound on Out-Sample Error

**Malik Magdon-Ismail**
Computer Science Department
Rensselaer Ploytechnic Institute
110 8th Street, Troy, NY 12180, USA
magdon@cs.rpi.edu

## Abstract

We define a data dependent permutation complexity for a hypothesis set $\mathcal{H}$, which is similar to a Rademacher complexity or maximum discrepancy. The permutation complexity is based (like the maximum discrepancy) on dependent sampling. We prove a uniform bound on the generalization error, as well as a concentration result which means that the permutation estimate can be efficiently estimated.

## 1   Introduction

Assume a standard setting with data $D = \{(\mathbf{x}_i, y_i)\}_{i=1}^n$, where $(\mathbf{x}_i, y_i)$ are sampled *iid* from the joint distribution $p(\mathbf{x}, y)$ on $\mathbb{R}^d \times \{\pm 1\}$. Let $\mathcal{H} = \{h : \mathbb{R}^d \mapsto \{\pm 1\}\}$ be a learning model which produces a hypothesis $g \in \mathcal{H}$ when given $D$ (we use $g$ for the hypothesis returned by the learning algorithm and $h$ for a generic hypothesis in $\mathcal{H}$). We assume the 0-1 loss, so the in-sample error is $e_{\text{in}}(h) = \frac{1}{2n} \sum_{i=1}^n (1 - y_i h(\mathbf{x}_i))$. The out-sample error $e_{\text{out}}(h) = \frac{1}{2} \mathbb{E}\left[(1 - yh(\mathbf{x}))\right]$; the expectation is over the joint distribution $p(\mathbf{x}, y)$. We wish to bound $e_{\text{out}}(g)$. To do so, we will bound $|e_{\text{out}}(h) - e_{\text{in}}(h)|$ uniformly over $\mathcal{H}$ for all distributions $p(\mathbf{x}, y)$; however, the bound itself will depend on the data, and hence the distribution. The classic distribution independent bound is the VC-bound (Vapnik and Chervonenkis, 1971); the hope is that by taking into account the data one can get a tighter bound. The data dependent *permutation complexity*[1] for $\mathcal{H}$ is defined by:

$$\mathcal{P}_{\mathcal{H}}(n, D) = \mathbb{E}_{\boldsymbol{\pi}}\left[\max_{h \in \mathcal{H}} \frac{1}{n} \sum_{i=1}^n y_{\pi_i} h(\mathbf{x}_i)\right].$$

Here, $\boldsymbol{\pi}$ is a uniformly random permutation on $\{1, \ldots, n\}$. $\mathcal{P}_{\mathcal{H}}(n, D)$ is an intuitively plausible measure of the complexity of a model, measuring its ability to correlate with a random permutation of the target values. The difficulty in analyzing $\mathcal{P}_{\mathcal{H}}$ is that $\{y_{\boldsymbol{\pi}_i}\}$ is an ordered random sample from $\mathbf{y} = [y_1, \ldots, y_n]$, sampled *without* replacement; as such it is a dependent sampling from a data driven distribution. Analogously, we may define the *bootstrap complexity*, using the bootstrap distribution $B$ on $\mathbf{y}$, where each sample $y_i^B$ is independent and uniformly random over $y_1, \ldots, y_n$:

$$\mathcal{B}_{\mathcal{H}}(n, D) = \mathbb{E}_B\left[\max_{h \in \mathcal{H}} \frac{1}{n} \sum_{i=1}^n y_i^B h(\mathbf{x}_i)\right].$$

When the average $y$-value $\bar{y} = 0$, the bootstrap complexity is exactly the Rademacher complexity (Bartlett and Mendelson, 2002; Fromont, 2007; Kääriäinen and Elomaa, 2003; Koltchinskii, 2001; Koltchinskii and Panchenko, 2000; Lozano, 2000; Lugosi and Nobel, 1999; Massart, 2000):

$$\mathcal{R}_{\mathcal{H}}(n, D) = \mathbb{E}_{\mathbf{r}}\left[\max_{h \in \mathcal{H}} \frac{1}{n} \sum_{i=1}^n r_i h(\mathbf{x}_i)\right],$$

where $\mathbf{r}$ is a random vector of i.i.d. fair $\pm 1$'s. The maximum discrepancy complexity measure $\Delta_{\mathcal{H}}(n, D)$ is similar to the Rademacher complexity, with the expectation over $\mathbf{r}$ being restricted to those $\mathbf{r}$ satisfying $\sum_{i=1}^{n} r_i = 0$,

$$\Delta_{\mathcal{H}}(n, D) = \mathbb{E}_{\mathbf{r}} \left[ \max_{h \in H} \frac{1}{n} \sum_{i=1}^{n} r_i y_i h(x_i) \right].$$

When $\bar{y} = 0$, the permutation complexity is the maximum discrepancy; the permutation complexity is to maximum discrepancy as the bootstrap complexity is to the Rademacher complexity. The permutation complexity maintains a little more information regarding the distribution. Indeed we prove a uniform bound very similar to the uniform bound obtained using the Rademacher complexity:

**Theorem 1** *With probability at least $1 - \delta$, for every $h \in \mathcal{H}$,*

$$e_{\text{out}}(h) \leq e_{\text{in}}(h) + \mathcal{P}_{\mathcal{H}}(n, D) + 13 \sqrt{\frac{1}{2n} \ln \frac{6}{\delta}}.$$

The probability in this theorem is with respect to the data distribution. The challenge in proving this theorem is to accomodate samples $(y_{\pi_i})$ constructed according to the data, and in a dependent way. Using our same proof technique, one can also obtain a similar uniform bound with the bootstrap complexity, where the samples are independent, but according to the data. The proof starts with the standard ghost sample and symmetrization argument. We then need to handle the data dependent sampling in the complexity measure, and this is done by introducing a *second ghost data set* to govern the sampling. The crucial aspect about sampling according to a second ghost data set is that the samples are now independent of the data; this is acceptable, provided the two methods of sampling are close enough; this is what constitutes the meat of the proof given in Section 2.2.

For a given permutation $\boldsymbol{\pi}$, one can compute $\max_{h \in \mathcal{H}} \frac{1}{n} \sum_{i=1}^{n} y_{\pi_i} h(\mathbf{x}_i)$ using an empirical risk minimization; however, the computation of the expectation over permutations is an exponential task, which needless to say is not feasible. Fortunately, we can establish that the permutation complexity is concentrated around its expectation, which means that in principle a single permutation suffices to compute the permutation complexity. Let $\boldsymbol{\pi}$ be a single random permutation.

**Theorem 2** *For an absolute constant $c \leq 6 + \sqrt{2/\ln 2}$, with probability at least $1 - \delta$,*

$$\mathcal{P}_{\mathcal{H}}(n, D) \leq \sup_{h \in \mathcal{H}} \frac{1}{n} \sum_{i=1}^{n} y_{\pi_i} h(\mathbf{x}_i) + c \sqrt{\frac{1}{2n} \ln \frac{3}{\delta}}.$$

The probability here is with respect to random permutations (i.e., it holds for any data set). It is easy to show concentration for the bootstrap complexity about its expectation – this follows from Mc-Diarmid's inequality because the samples are independent. The complication with the permutation complexity is that the samples are not independent. Nevertheless, we can show the concentration indirectly by first relating the two complexities for any data set, and then using the concentration of the bootstrap complexity (see Section 2.3).

**Empirical Results.** For a single random permutation, with probability at least $1 - \delta$,

$$e_{\text{out}}(h) \leq e_{\text{in}}(h) + \sup_{h \in \mathcal{H}} \frac{1}{n} \sum_{i=1}^{n} y_{\pi_i} h(\mathbf{x}_i) + O \left( \sqrt{\frac{1}{n} \ln \frac{1}{\delta}} \right).$$

Asymptotically, one random permutation suffices; in practice, one should average over a few. Indeed, a permutation based validation estimate for model selection has been extensively tested (see Magdon-Ismail and Mertsalov (2010) for details); for classification, this permutation estimate is the permutation complexity after removing a bias term. It outperformed LOO-cross validation and the Rademacher complexity on real data. We restate those results here, comparing model selection using the permutation estimate versus using the Rademacher complexity (using real data sets from the UCI Machine Learning repository (Asuncion and Newman, 2007)). The performance metric is the regret when compared to oracle model selection on a held out set (lower regret is better). We considered two model selection tasks: choosing the number of leafs in a decision tree; and, selecting $k$ in the $k$-nearest neighbor method. The results reported here are averaged over several (10,000 or more) random splits of the data into a training set and held out set. We define a learning episode as an empirical risk minimization on $O(n)$ data points.

| Data | $n$ | 10 Learning Episodes | | | | 100 Learning Episodes | | | |
|------|-----|------|------|------|------|------|------|------|------|
| | | Decision Trees | | $k$-NN | | Decision Trees | | $k$-NN | |
| | | Perm. | Rad. | Perm. | Rad. | Perm. | Rad. | Perm. | Rad. |
| Abalone | 3,132 | 0.02 | 0.02 | **0.09** | 0.12 | 0.02 | 0.02 | 0.04 | 0.04 |
| Ionosphere | 263 | **0.18** | 0.19 | **0.75** | 0.84 | **0.16** | 0.17 | **0.70** | 0.83 |
| M.Mass | 667 | 0.06 | 0.06 | **0.11** | 0.12 | 0.05 | 0.05 | 0.11 | 0.11 |
| Parkinsons | 144 | **0.34** | 0.40 | **0.32** | 0.44 | **0.34** | 0.41 | **0.33** | 0.43 |
| Pima Ind. | 576 | 0.07 | 0.07 | **0.12** | 0.15 | 0.07 | 0.07 | **0.11** | 0.14 |
| Spambase | 3,450 | 0.07 | 0.07 | **0.43** | 0.54 | **0.06** | 0.07 | **0.43** | 0.55 |
| Transfusion | 561 | **0.08** | 0.09 | **0.12** | 0.19 | **0.08** | 0.09 | **0.12** | 0.19 |
| WDBC | 426 | **0.24** | 0.37 | **0.33** | 0.50 | **0.23** | 0.34 | **0.34** | 0.51 |
| Diffusion | 2,665 | 0.03 | **0.02** | 0.06 | **0.04** | 0.03 | **0.02** | 0.06 | **0.03** |

The permutation complexity appears to dominate most of the time (especially when $n$ is small); and, when it fails to dominate, it is as good or only slightly worse than the Rademacher estimate. It is not surprising that as $n$ increases, the performances of the various complexities converges. Asymptotically, one can deduce several relationships between them, for example the maximum discrepancy can be asymptotically bounded from above and below by the Rademacher complexity. Similarly, (see Lemma 5), the bootstrap and permutation complexities are equal, asymptotically. The small sample performance of the complexities as bounding tools is not easy to discern theoretically, which is where the empirics comes in. An intuition for why the permutation complexity performs relatively well is because it maintains more of the true data distribution. Indeed, the permutation method for validation was found to work well empirically, even in regression (Magdon-Ismail and Mertsalov, 2010); however, our permutation complexity bound only applies to classification.

**Open Questions.** Can the permutation complexity bound be extended beyond classification to (for example) regression with bounded loss? The permutation complexity displays a bias for severely unbalanced data; can this bias be removed. We conjecture that it should be possible to get a better uniform bound in terms of $\mathbb{E}_{\boldsymbol{\pi}}[\max_{h \in \mathcal{H}} \frac{1}{n} \sum_{i=1}^{n} (y_{\pi_i} - \bar{y}) h(\mathbf{x}_i)]$.

## 1.1 Related Work

Out-sample error estimation has extensive coverage, both in the statistics and learning commuities.

(i) *Statistical methods* try to estimate the out-sample error asymptotically in $n$, and give consistent estimates under certain model assumptions, for example: final prediction error (FPE) (Akaike, 1974); Generalized Cross Validation (GCV) (Craven and Wahba, 1979); or, covariance-type penalties (Efron, 2004; Wang and Shen, 2006). Statistical methods tend to work well when the model has been well specified. Such methods are not our primary focus.

(ii) *Sampling methods*, such as leave-one-out cross validation (LOO-CV), try to estimate the out-sample error directly. Cross validation is perhaps the most used validation method, dating as far back as 1931 (Larson, 1931; Wherry, 1931, 1951; Katzell, 1951; Cureton, 1951; Mosier, 1951; Stone, 1974). The permutation complexity uses a "sampled" data set on which to compute the complexity; other than this superficial similarity, the estimates are inherently different.

(iii) *Bounds*. The most celebrated uniform bound on generalization error is the distribution independent bound of Vapnik-Chervonenkis (VC-bound) (Vapnik and Chervonenkis, 1971). Since the VC-dimension may be hard to compute, empirical estimates have been suggested, (Vapnik *et al.*, 1994). The VC-bound is optimal among distribution independent bounds; however, for a particular distribution, it could be sub-optimal. Several data dependent bounds have already been mentioned, which can typically be estimated in-sample via optimization: maximum discrepancy (Bartlett *et al.*, 2002); Rademacher-style penalties (Bartlett and Mendelson, 2002; Fromont, 2007; Kääriäinen and Elomaa, 2003; Koltchinskii, 2001; Koltchinskii and Panchenko, 2000; Lozano, 2000; Lugosi and Nobel, 1999; Massart, 2000); margin based bounds, for example (Shawe-Taylor *et al.*, 1998). Generalizations to Gaussian and symmetric, bounded variance **r** have also been suggested, (Bartlett and Mendelson, 2002; Fromont, 2007) . One main application of such bounds is that any such approximate estimate of the out-sample error (which satisfies some bound of the form of the permutation complexity bound) can be used for model selection, after adding a (small) penalty for the "complex-

ity of model selection" (see Bartlett *et al.* (2002)). In practice, this penalty for the complexity of model selection is ignored (as in Bartlett *et al.* (2002)).

(iv) *Permutation Methods* are not new to statistics (Good, 2005; Golland *et al.*, 2005; Wiklund *et al.*, 2007). Golland *et al.* (2005) show concentration for a permutation based test of significance for the improved performance of a more complex model, using the Rademacher complexity. We directly give a uniform bound for the out-sample error in terms of a permutation complexity, answering a question posed in (Golland *et al.*, 2005) which asks whether there is a direct link between permutation statistics and generalization errors. Indeed, Magdon-Ismail and Mertsalov (2010) construct a permutation estimate for validation which they empirically test in both classification and regression problems. For classification, their estimate is related to the permutation complexity.

Most relevant to this work are Rademacher penalties and the corresponding (sampling without replacement) maximum discrepancy. Bartlett *et al.* (2002) give a uniform bound using the maximum discrepancy which is in some sense a uniform bound based on a sampling without replacement (dependent sampling); however, the sampling distribution is fixed, independent of the data. It is illustrative to briefly sketch the derivation of the maximum discrepancy bound. Adapting the proof in Bartlett *et al.* (2002) and ignoring terms which are $O\left(\left(\frac{1}{n}\ln\frac{1}{\delta}\right)^{1/2}\right)$, with probability at least $1-\delta$:

$$
\begin{aligned}
e_{\text{out}}(h) \quad &\leq \quad e_{\text{in}}(h) + \sup_{h\in\mathcal{H}}\left\{e_{\text{out}}(h) - e_{\text{in}}(h)\right\} \overset{(a)}{\leq} e_{\text{in}}(h) + \mathbb{E}_D \sup_{h\in\mathcal{H}}\left\{e_{\text{out}}(h) - e_{\text{in}}(h)\right\}, \\
&\overset{(b)}{=} \quad e_{\text{in}}(h) + \mathbb{E}_D \sup_{h\in\mathcal{H}}\left\{\mathbb{E}_{D'} \frac{1}{2n}\sum_{i=1}^{n} y_i h(\mathbf{x}_i) - y_i' h(\mathbf{x}_i')\right\}, \\
&\overset{(c)}{\leq} \quad e_{\text{in}}(h) + \mathbb{E}_{D,D'} \max_{h\in\mathcal{H}}\left\{\frac{1}{2n}\sum_{i=1}^{n} y_i h(\mathbf{x}_i) - y_i' h(\mathbf{x}_i')\right\}, \\
&\overset{(d)}{\leq} \quad e_{\text{in}}(h) + \mathbb{E}_{D,D'} \max_{h\in\mathcal{H}}\left\{\frac{1}{n}\sum_{i=1}^{n/2} y_i h(\mathbf{x}_i) - y_i' h(\mathbf{x}_i')\right\}, \\
&= \quad e_{\text{in}}(h) + \mathbb{E}_D \Delta_{\mathcal{H}}(n, D) \overset{(e)}{\leq} e_{\text{in}}(h) + \Delta_{\mathcal{H}}(n, D),
\end{aligned}
$$

(a) follows from McDiarmid's inequality because $e_{\text{out}}(h) - e_{\text{in}}(h)$ is stable to a single point perturbation for every $h$, hence the supremum is also stable; in (b) appears a ghost data set and (c) follows by convexity of the supremum; in (d), we break the sum into two equal parts, which adds the factor of two; finally, (e) follows again by McDiarmid's inequality because $\Delta_{\mathcal{H}}$ is stable to single point perturbations. The discrepancy automatically drops out from using the ghost sample; this does not happen with data dependent permutation sampling, which is where the difficulty lies.

## 2 Permutation Complexity Uniform Bound

We now give the proof of Theorem 1. We will adapt the standard ghost sample approach in VC-type proofs and the symmetrization trick in (Giné and Zinn, 1984) which has greatly simplified VC-style proofs. In general, high probability results are with respect to the distribution over data sets. Our main bounding tool will be McDiarmid's inequality:

**Lemma 1 (McDiarmid (1989))** *Let $X_i \in A_i$ be independent; suppose $f : \prod_i A_i \mapsto \mathbb{R}$ satisfies*

$$
\sup_{\substack{(x_1,\ldots,x_n)\in\prod_i A_i \\ z\in A_j}} |f(\mathbf{x}) - f(x_1,\ldots,x_{j-1},z,x_{j+1},\ldots,x_n)| \leq c_j,
$$

*for $j = 1,\ldots,n$. Then, with probability at least $1 - \delta$,*

$$
f(X_1,\ldots,X_n) \leq \mathbb{E}f(X_1,\ldots,X_n) + \sqrt{\frac{1}{2}\sum_{i=1}^{n} c_i^2 \ln\frac{1}{\delta}}.
$$

We also obtain $\mathbb{E}f \leq f + \sqrt{\frac{1}{2}\sum_{i=1}^{n} c_i^2 \ln\frac{1}{\delta}}$ by using $-f$ in McDiarmid's inequality.

## 2.1 Permutation Complexity

The out-sample permutation complexity of a model is:

$$\mathcal{P}_{\mathcal{H}}(n) = \mathbb{E}_D \mathcal{P}_{\mathcal{H}}(n, D) = \mathbb{E}_{D, \boldsymbol{\pi}} \left[ \max_{h \in \mathcal{H}} \frac{1}{n} \sum_{i=1}^{n} y_{\pi_i} h(\mathbf{x}_i) \right],$$

where the expectation is over the data $D = (\mathbf{x}_1, y_1), \ldots, (\mathbf{x}_n, y_n)$ and a random permutation $\boldsymbol{\pi}$. Let $D'$ differ from $D$ only in one example, $(\mathbf{x}_j, y_j) \to (\mathbf{x}'_j, y'_j)$.

**Lemma 2** $|\mathcal{P}_{\mathcal{H}}(n, D) - \mathcal{P}_{\mathcal{H}}(n, D')| \leq \frac{4}{n}$.

**Proof:** For any permutation $\boldsymbol{\pi}$ and every $h \in \mathcal{H}$, the sum $\sum_{i=1}^{n} y_{\pi_i} h(\mathbf{x}_i)$ changes by at most 4 in going from $D$ to $D'$; thus, the maximum over $h \in \mathcal{H}$ changes by at most 4. ∎

Lemma 2 together with McDiarmid's inequality implies a concentration of $\mathcal{P}_{\mathcal{H}}(n, D)$ about $\mathcal{P}_{\mathcal{H}}(n)$, which means we can work with $\mathcal{P}_{\mathcal{H}}(n, D)$ instead of the unknown $\mathcal{P}_{\mathcal{H}}(n)$.

**Corollary 1** *With probability at least* $1 - \delta$, $\mathcal{P}_{\mathcal{H}}(n) \leq \mathcal{P}_{\mathcal{H}}(n, D) + 4\sqrt{\frac{1}{2n} \ln \frac{1}{\delta}}$.

Since $e_{\text{in}}(h) = \frac{1}{2}(1 - \frac{1}{n} \sum_{i=1}^{n} y_i h(\mathbf{x}_i))$, the empirical risk minimizer $g^{\boldsymbol{\pi}}$ on the permuted targets $\mathbf{y}^{\boldsymbol{\pi}}$ can be used to compute $\mathcal{P}_{\mathcal{H}}(n, D)$ for a particular permutation $\boldsymbol{\pi}$.

## 2.2 Bounding the Out-Sample Error

To bound $\sup_{h \in \mathcal{H}}\{e_{\text{out}}(h) - e_{\text{in}}(h)\}$, we first use the standard ghost sample and symmetrization arguments typical of modern generalization error proofs (see for example Bartlett and Mendelson (2002); Shawe-Taylor and Cristianini (2004)). Let $\mathbf{r}'' = [r''_1, \ldots, r''_n]$ be a $\pm 1$ sequence.

**Lemma 3** *With probability at least* $1 - \delta$:

$$\sup_{h \in \mathcal{H}}\{e_{\text{out}}(h) - e_{\text{in}}(h)\} \quad \leq \quad \mathbb{E}_{D, D'} \left[ \sup_{h \in \mathcal{H}} \left\{ \frac{1}{2n} \sum_{i=1}^{n} r''_i (y_i h(\mathbf{x}_i) - y'_i h(\mathbf{x}'_i)) \right\} \right] + \sqrt{\frac{1}{2n} \ln \frac{1}{\delta}}.$$

**Proof:** We proceed as in the proof of the maximum discrepancy bound in Section 1.1:

$$\sup_{h \in \mathcal{H}}\{e_{\text{out}}(h) - e_{\text{in}}(h)\} \overset{(a)}{\leq} \mathbb{E}_{D, D'} \left[ \sup_{h \in \mathcal{H}} \left\{ \frac{1}{2n} \sum_{i=1}^{n} y_i h(\mathbf{x}_i) - y'_i h(\mathbf{x}'_i) \right\} \right] + \sqrt{\frac{1}{2n} \ln \frac{1}{\delta}},$$

$$\overset{(b)}{=} \mathbb{E}_{D, D'} \left[ \sup_{h \in \mathcal{H}} \left\{ \frac{1}{2n} \sum_{i=1}^{n} r''_i (y_i h(\mathbf{x}_i) - y'_i h(\mathbf{x}'_i)) \right\} \right] + \sqrt{\frac{1}{2n} \ln \frac{1}{\delta}}.$$

In (a), the $O((\frac{1}{n} \ln \frac{1}{\delta})^{1/2})$ term is from applying McDiarmid's inequality because $e_{\text{in}}(h)$ changes by at most $\frac{1}{n}$ if one data point changes, and so the supremum changes by at most that much; (b) follows because $r''_i = -1$ corresponds to exchanging $\mathbf{x}_i, \mathbf{x}'_i$ in the expectation which does not change the expectation (it amounts to relabeling of random variables). ∎

Lemma 3 holds for an *arbitrary* sequence $\mathbf{r}''$ which is independent of $D, D'$; we can take the expectation with respect to $\mathbf{r}''$, for *arbitrarily* distributed $\mathbf{r}''$, as long as $\mathbf{r}''$ is independent of $D, D'$.

### 2.2.1 Generating Permutations with $\pm 1$ Sequences

Fix $\mathbf{y}$; for a given permutation $\boldsymbol{\pi}$, define a corresponding $\pm 1$ sequence $\mathbf{r}^{\boldsymbol{\pi}}$ by $r_i^{\boldsymbol{\pi}} = y_{\boldsymbol{\pi}_i} y_i$; then, $y_{\boldsymbol{\pi}_i} = r_i^{\boldsymbol{\pi}} y_i$. Thus, given $\mathbf{y}$, for each of the $n!$ permutations $\boldsymbol{\pi}_1, \ldots, \boldsymbol{\pi}_{n!}$, we have a corresponding $\pm 1$ sequence $\mathbf{r}^{\boldsymbol{\pi}_i}$; we thus obtain a multiset of sequences $S_{\mathbf{y}} = \{\mathbf{r}^{\boldsymbol{\pi}_1}, \ldots, \mathbf{r}^{\boldsymbol{\pi}_{n!}}\}$ (there may be repetitions as two different permutations may result in the same sequence of $\pm 1$ values); we thus have a mapping from permutations to the $\pm 1$ sequences in $S_{\mathbf{y}}$. If $\mathbf{r}$, a random vector of $\pm 1$s, is

uniform on $S_{\mathbf{y}}$, then $\mathbf{r}.\mathbf{y}$ (componentwise product) is uniform over the permutations of $\mathbf{y}$. We say that $S_{\mathbf{y}}$ generates the permutations on $\mathbf{y}$. Similarly, we can define $S_{\mathbf{y}'}$, the generator of permutations on $\mathbf{y}'$. Unfortunately, $S_{\mathbf{y}}, S_{\mathbf{y}'}$ depend on $D, D'$, and so we can't take the expectation uniformly over (for example) $\mathbf{r} \in S_{\mathbf{y}}$. We can overcome this by introducing a second ghost sample $D''$ to "approximately" generate the permutations for $\mathbf{y}, \mathbf{y}'$, ultimately allowing us to prove the main result.

**Theorem 3** *With probability at least $1 - 5\delta$,*

$$\sup_{h \in \mathcal{H}} \{e_{\text{out}}(h) - e_{\text{in}}(h)\} \leq \mathcal{P}_{\mathcal{H}}(n) + 9\sqrt{\frac{1}{2n} \ln \frac{1}{\delta}},$$

We obtain Theorem 1 by combining Theorem 3 with Corollary 1.

### 2.2.2 Proof of Theorem 3

Let $D''$ be a *second, independent* ghost sample, and $S_{\mathbf{y}''}$ the generator of permutations for $\mathbf{y}''$. In Lemma 3, take the expectation over $\mathbf{r}''$ uniform on $S_{\mathbf{y}''}$. The first term on the RHS becomes

$$\mathbb{E}_{D,D',D''} \frac{1}{n!} \sum_{\boldsymbol{\pi}} \left[ \sup_{h \in \mathcal{H}} \frac{1}{2n} \sum_{i=1}^{n} r_i''(\boldsymbol{\pi})(y_i h(\mathbf{x}_i) - y_i' h(\mathbf{x}_i')) \right], \qquad (1)$$

where each permutation $\boldsymbol{\pi}$ induces a particular sequence $\mathbf{r}''(\boldsymbol{\pi}) \in S_{\mathbf{y}''}$ (previously we used $r_i^{\boldsymbol{\pi}}$ which is now $r_i(\boldsymbol{\pi})$). Consider the sequences $\mathbf{r}, \mathbf{r}'$ corresponding to the permutations on $\mathbf{y}$ and $\mathbf{y}'$. The next lemma will ultimately relate the expectation over permutations in the second ghost data set to the permutations over $D, D'$.

**Lemma 4** *With probability at least $1 - 2\delta$, there is a one-to-one mapping from the sequences in $S_{\mathbf{y}''} = \{\mathbf{r}''(\boldsymbol{\pi})\}_{\boldsymbol{\pi}}$ to $S_{\mathbf{y}} = \{\mathbf{r}(\boldsymbol{\pi})\}_{\boldsymbol{\pi}}$ such that*

$$\left| \frac{1}{2n} \sum_{i=1}^{n} (r_i'' - r_i(\mathbf{r}'')) y_i h(\mathbf{x}_i) \right| \leq \sqrt{\frac{8}{n} \ln \frac{1}{\delta}},$$

*for every $\mathbf{r}'' \in S_{\mathbf{y}''}$ and every $h \in \mathcal{H}$ (we write $\mathbf{r}(\mathbf{r}'')$ to denote the sequence $\mathbf{r} \in S_{\mathbf{y}}$ to which $\mathbf{r}''$ is mapped). Similarly, there exists such a mapping from $S_{\mathbf{y}''}$ to $S_{\mathbf{y}'}$.*

The probability here is with respect to $\mathbf{y}$, $\mathbf{y}'$ and $\mathbf{y}''$. This lemma says that the permutation generating sets $S_{\mathbf{y}''}$, $S_{\mathbf{y}'}$, and $S_{\mathbf{y}}$ are essentially equivalent.

**Proof:** We can (without loss of generality) reorder the points in $D''$ so that the first $k''$ are $+1$, so $y_1'' = \cdots = y_{k''}'' = +1$, and the remaining are $-1$. Similarly, we can order the points in $D$ so that the first $k$ are $+1$, so $y_1 = \cdots = y_k = +1$. We now construct the mapping from $S_{\mathbf{y}''}$ to $S_{\mathbf{y}}$ as follows. For a given permutation $\boldsymbol{\pi}$, we map $\mathbf{r}''(\pi) \in S_{\mathbf{y}''}$ to $\mathbf{r}(\pi) \in S_{\mathbf{y}}$. This mapping is clearly bijective since every permutation corresponds uniquely to a sequence in $S_{\mathbf{y}}$ (and $S_{\mathbf{y}''}$).

Let $r_i = y_{\pi_i} y_i$ and $r_i'' = y_{\pi_i}'' y_i''$. If $r_i \neq r_i''$, either $y_{\pi_i} \neq y_{\pi_i}''$ or $y_i \neq y_i''$. Since $\mathbf{y}$ and $\mathbf{y}''$ disagree on exactly $|k - k''|$ locations (and similarly for $y_{\boldsymbol{\pi}}$ and $y_{\boldsymbol{\pi}}''$), the number of locations where $\mathbf{r}$ and $\mathbf{r}''$ disagree is therefore at most $2|k - k''|$. Thus, for any $\mathbf{r}''$ and any $h \in \mathcal{H}$,

$$\left| \frac{1}{2n} \sum_{i=1}^{n} (r_i'' - r_i(\mathbf{r}'')) y_i h(\mathbf{x}_i) \right| \leq \frac{1}{2n} \sum_{i=1}^{n} |r_i'' - r_i(\mathbf{r}'')| \, |y_i h(\mathbf{x}_i)|$$

$$= \frac{1}{2n} \sum_{i=1}^{n} |r_i'' - r_i(\mathbf{r}'')| \leq \frac{2|k - k''|}{n}.$$

We observe that $\sum_{i=1}^{n} (y_i - y_i'') = 2(k - k'')$ and so,

$$\left| \frac{1}{2n} \sum_{i=1}^{n} (r_i'' - r_i(\mathbf{r}'')) y_i h(\mathbf{x}_i) \right| \leq \left| \frac{1}{n} \sum_{i=1}^{n} (y_i - y_i'') \right| = \left| \frac{1}{n} \sum_{i=1}^{n} z_i \right|,$$

where $z_i = y_i - y_i''$. Since $\mathbf{y}$ and $\mathbf{y}''$ are identically distributed, $z_i$ are independent and zero mean. We consider the function $f(z_1, \ldots, z_n) = \frac{1}{n} \sum_{i=1}^{n} z_i$. Since $z_i \in \{0, \pm 2\}$, if you change one of the

$z_i$, $f$ changes by at most $\frac{4}{n}$, and so the conditions hold to apply McDiarmid's inequality to $f$. Thus, using the symmetry of $z_i$, with probability at least $1 - 2\delta$, $\left|\frac{8}{n}\sum_{i=1}^{n}z_i\right| \leq \sqrt{\frac{1}{2n}\ln\frac{1}{\delta}}$. ∎

Given $D, D', D''$, assume the mappings which are known to exist by the previous lemma are $\mathbf{r}(\mathbf{r}'')$ and $\mathbf{r}'(\mathbf{r}'')$. We can rewrite the internal summand in the expression of Equation (1) using the equality

$$r_i''(y_ih(\mathbf{x}_i) - y_i'h(\mathbf{x}_i')) = (r_i'' - r_i(\mathbf{r}'') + r_i(\mathbf{r}''))y_ih(\mathbf{x}_i) - (r_i'' - r_i'(\mathbf{r}'') + r_i'(\mathbf{r}''))y_i'h(\mathbf{x}_i').$$

Using Lemma 4, we can, with probability at least $1 - 2\delta$, bound the term which involves $(r_i'' - r_i(\mathbf{r}''))$ in Equation (1); and, similarly, with probability at least $1 - 2\delta$, we bound the term involving $(r_i'' - r_i'(\mathbf{r}''))$. Thus, with probability at least $1 - 4\delta$, the expression in Equation (1) is bounded by:

$$\mathbb{E}_{D,D',D''}\frac{1}{n!}\sum_{\boldsymbol{\pi}}\left[\sup_{h\in\mathcal{H}}\frac{1}{2n}\sum_{i=1}^{n}(r_i(\mathbf{r}'')y_ih(\mathbf{x}_i) - r_i'(\mathbf{r}'')y_i'h(\mathbf{x}_i'))\right] + 2\sqrt{\frac{8}{n}\ln\frac{1}{\delta}},$$

where $\mathbf{r}''(\boldsymbol{\pi})$ cycles through the sequences in $S_{\mathbf{y}''}$. Since the mappings $\mathbf{r}(\mathbf{r}'')$ and $\mathbf{r}'(\mathbf{r}'')$ are one-to-one, $\mathbf{r}(\mathbf{r}'').\mathbf{y}$ cycles through the permutations of $\mathbf{y}$, and similarly for $\mathbf{r}'(\mathbf{r}'').\mathbf{y}'$. Since $\mathcal{H}$ is closed under negation, we finally obtain the bound

$$\mathbb{E}_D\frac{1}{n!}\sum_{\boldsymbol{\pi}}\left[\sup_{h\in\mathcal{H}}\frac{1}{2n}\sum_{i=1}^{n}y_{\boldsymbol{\pi}_i}h(\mathbf{x}_i)\right] + \mathbb{E}_{D'}\frac{1}{n!}\sum_{\boldsymbol{\pi}}\left[\sup_{h\in\mathcal{H}}\frac{1}{2n}\sum_{i=1}^{n}y_{\boldsymbol{\pi}_i}'h(\mathbf{x}_i')\right] + 2\sqrt{\frac{8}{n}\ln\frac{1}{\delta}};$$

Using this in Lemma 3, with probability at least $1 - 5\delta$,

$$\sup_{h\in\mathcal{H}}\{e_{\text{out}}(h) - e_{\text{in}}(h)\} \leq \mathcal{P}_{\mathcal{H}}(n) + 9\sqrt{\frac{1}{2n}\ln\frac{1}{\delta}}.$$

∎

**Commentary.** (i) The permutation complexity bound needs empirical risk minimization, which is notoriously hard; however, if the *same* algorithm is used for learning as well as computing $\mathcal{P}$, we can view it as optimization over a constrained hypothesis set (this is especially so with regularization); the bounds now hold. (ii) The same proof technique can be used to get a bootstrap complexity bound; the result is similar. (iii) One could bound $\mathcal{P}_{\mathcal{H}}$ for VC function classes, showing that this data dependent bound is asymptotically no worse than a VC-type bound. Bounding permutation complexity on specific domains could follow the methods in Bartlett and Mendelson (2002).

### 2.3 Estimating $\mathcal{P}_{\mathcal{H}}(n, D)$ Using a Single Permutation

We now prove Theorem 2, which states that one can essentially estimate $\mathcal{P}_{\mathcal{H}}(n, D)$ (an average over all permutations) by $\sup_{h\in\mathcal{H}}\frac{1}{n}\sum_{i=1}^{n}y_{\pi_i}h(\mathbf{x}_i)$, using just a single randomly selected permutation $\boldsymbol{\pi}$. Our proof is indirect: we will link $\mathcal{P}_{\mathcal{H}}$ to the bootstrap complexity $\mathcal{B}_{\mathcal{H}}$. The bootstrap complexity is concentrated via an easy application of McDiarmid's inequality, which will ultimately allow us to conclude that the permutation estimate is also concentrated. The bootstrap distribution $B$ constructs a random sequence $\mathbf{y}^B$ of $n$ independent uniform samples from $y_1, \ldots, y_n$; the key requirement is that $y_i^B$ are independent samples. There are $n^n$ (not distinct) possible bootstrap sequences.

**Lemma 5** $|\mathcal{B}_{\mathcal{H}}(n, D) - \mathcal{P}_{\mathcal{H}}(n, D)| \leq \frac{1}{\sqrt{n}}$.

**Proof:** Let $k$ be the number of $y_i$ which are $+1$; we condition on $\kappa$, the number of $+1$ in the bootstrap sample. Suppose $B|\kappa$ samples uniformly among all sequences with $\kappa$ entries being $+1$.

$$\mathcal{B}_{\mathcal{H}}(n, D) = \mathbb{E}_{\kappa}\,\mathbb{E}_{B|\kappa}\left[\sup_{h\in\mathcal{H}}\frac{1}{n}\sum_{i=1}^{n}y_i^Bh(\mathbf{x}_i)\,\bigg|\,\kappa\right],$$

The key observation is that we can generate all samples uniformly according to $B|\kappa$ by first generating a random permutation and then selecting randomly $|k - \kappa|$ $+1$'s (or $-1$'s) to flip, so:

$$\mathbb{E}_{B|\kappa}\left[\sup_{h\in\mathcal{H}}\frac{1}{n}\sum_{i=1}^{n}y_i^Bh(\mathbf{x}_i)\,\bigg|\,\kappa\right] = \mathbb{E}_{F_{|k-\kappa|}}\,\mathbb{E}_{\boldsymbol{\pi}}\left[\sup_{h\in\mathcal{H}}\frac{1}{n}\sum_{i=1}^{n}y_{\boldsymbol{\pi}_i}^Fh(\mathbf{x}_i)\right].$$

($F$ denotes the flipping random process.) Since $y_{\boldsymbol{\pi}_i}^F$ differs from $y_{\boldsymbol{\pi}_i}$ in exactly $|k - \kappa|$ positions,

$$\sup_{h \in \mathcal{H}} \frac{1}{n} \sum_{i=1}^{n} y_{\boldsymbol{\pi}_i} h(\mathbf{x}_i) - \frac{2|k - \kappa|}{n} \leq \sup_{h \in \mathcal{H}} \frac{1}{n} \sum_{i=1}^{n} y_{\boldsymbol{\pi}_i}^F h(\mathbf{x}_i) \leq \sup_{h \in \mathcal{H}} \frac{1}{n} \sum_{i=1}^{n} y_{\boldsymbol{\pi}_i} h(\mathbf{x}_i) + \frac{2|k - \kappa|}{n}.$$

Thus,

$$|\mathcal{B}_{\mathcal{H}}(n, D) - \mathcal{P}_{\mathcal{H}}(n, D)| \leq \frac{2}{n} \mathbb{E}_\kappa \left[ |k - \kappa| \right].$$

Since $\mathbb{E}_\kappa[|k - \kappa|] \leq \sqrt{\mathrm{Var}[k - \kappa]} \leq \frac{1}{2}\sqrt{n}$ (because $\kappa$ is binomial), the result follows. ∎

In addition to furthering our cause toward the proof of Theorem 2, Lemma 5 is interesting in its own right, because it says that permutation and bootstrap sampling are asymptotically similar. The nice thing about the bootstrap estimate is that the expectation is over independent $y_1^B, \ldots, y_n^B$. Since the bootstrap complexity changes by at most $\frac{2}{n}$ if you change one sample, by McDiarmid's inequality,

**Lemma 6** *For a random bootstrap sample B, with probability at least $1 - \delta$,*

$$\mathcal{B}_{\mathcal{H}}(n, D) \leq \sup_{h \in \mathcal{H}} \frac{1}{n} \sum_{i=1}^{n} y_i^B h(\mathbf{x}_i) + 2\sqrt{\frac{1}{2n} \ln \frac{1}{\delta}}.$$

We now prove concentration for estimating $\mathcal{P}_{\mathcal{H}}(n, D)$. As in the proof of Lemma 5, generate $\mathbf{y}^B$ in two steps. First generate $\kappa$, the number of $+1$'s in $\mathbf{y}^B$; $\kappa$ is binomial. Now, generate a random permutation $\mathbf{y}^{\boldsymbol{\pi}}$, and flip (as appropriate) a randomly selected $|k - \kappa|$ entries, where $k$ is the number of $+1$'s in $\mathbf{y}$. If we apply McDiarmid's inequality to the function which equals the number of $+1$'s, we immediately get that with probability at least $1 - 2\delta$, $|\kappa - k| \leq (\frac{1}{2}n \ln \frac{1}{\delta})^{1/2}$. Thus, with probability at least $1 - 2\delta$, $\mathbf{y}^B$ differs from $\mathbf{y}^{\boldsymbol{\pi}}$ in at most $(2n \ln \frac{1}{\delta})^{1/2}$ positions. Each flip changes the complexity by at most 2, hence, with probability at least $1 - 2\delta$,

$$\sup_{h \in \mathcal{H}} \frac{1}{n} \sum_{i=1}^{n} y_i^B h(\mathbf{x}_i) \leq \sup_{h \in \mathcal{H}} \frac{1}{n} \sum_{i=1}^{n} y_{\boldsymbol{\pi}_i} h(\mathbf{x}_i) + 4\sqrt{\frac{1}{2n} \ln \frac{1}{\delta}}.$$

We conclude that for a random permutation $\boldsymbol{\pi}$, with probability at least $1 - 3\delta$,

$$\mathcal{B}_{\mathcal{H}}(n, D) \leq \sup_{h \in \mathcal{H}} \frac{1}{n} \sum_{i=1}^{n} y_{\boldsymbol{\pi}_i} h(\mathbf{x}_i) + 6\sqrt{\frac{1}{2n} \ln \frac{1}{\delta}}.$$

Now, combining with Lemma 5, we obtain Theorem 2 after a little algebra, because $\delta < 1$. ∎

We have not only established that $\mathcal{P}_{\mathcal{H}}$ is concentrated, but we have also established a general connection between the permutation and bootstrap based estimates. In this particular case, we see that sampling with and without replacement are very closely related. In practice, sampling without replacement can be very different, because one is never in the truly asymptotic regime. Along that vein, even though we have concentration, it pays to take the average over a few permutations.

## Footnotes

[1] For simplicity, we assume that $\mathcal{H}$ is closed under negation; generally, all the results hold with the complexities defined using absolute values, so for example $\mathcal{P}_{\mathcal{H}}(n, D) = \mathbb{E}_{\boldsymbol{\pi}}\left[\max_{h \in \mathcal{H}} \left|\frac{1}{n} \sum_{i=1}^n y_{\pi_i} h(\mathbf{x}_i)\right|\right]$.

## References

Akaike, H. (1974). A new look at the statistical model identification. *IEEE Trans. Aut. Cont.*, **19**, 716–723.

Asuncion, A. and Newman, D. (2007). UCI machine learning repository.

Bartlett, P. L. and Mendelson, S. (2002). Rademacher and Gaussian complexities: Risk bounds and structural results. *Journal of Machine Learning Research*, **3**, 463–482.

Bartlett, P. L., Boucheron, S., and Lugosi, G. (2002). Model selection and error estimation. *Machine Learning*, **48**, 85–113.

Craven, P. and Wahba, G. (1979). Smoothing noisy data with spline functions. *Numerische Mathematik*, **31**, 377–403.

Cureton, E. E. (1951). Symposium: The need and means of cross-validation: II approximate linear restraints and best predictor weights. *Education and Psychology Measurement*, **11**, 12–15.

Efron, B. (2004). The estimation of prediction error: Covariance penalties and cross-validation. *Journal of the American Statistical Association*, **99**(467), 619–632.

Fromont, M. (2007). Model selection by bootstrap penalization for classification. *Machine Learning*, **66**(2-3), 165–207.

Giné, E. and Zinn, J. (1984). Some limit theorems for empirical processes. *Annals of Prob.*, **12**, 929–989.

Golland, P., Liang, F., Mukherjee, S., and Panchenko, D. (2005). Permutation tests for classification. *Learning Theory*, pages 501–515.

Good, P. (2005). *Permutation, parametric, and bootstrap tests of hypotheses*. Springer.

Kääriäinen, M. and Elomaa, T. (2003). Rademacher penalization over decision tree prunings. In *In Proc. 14th European Conference on Machine Learning*, pages 193–204.

Katzell, R. A. (1951). Symposium: The need and means of cross-validation: III cross validation of item analyses. *Education and Psychology Measurement*, **11**, 16–22.

Koltchinskii, V. (2001). Rademacher penalties and structural risk minimization. *IEEE Transactions on Information Theory*, **47**(5), 1902–1914.

Koltchinskii, V. and Panchenko, D. (2000). Rademacher processes and bounding the risk of function learning. In E. Gine, D. Mason, and J. Wellner, editors, *High Dimensional Prob. II*, volume 47, pages 443–459.

Larson, S. C. (1931). The shrinkage of the coefficient of multiple correlation. *Journal of Education Psychology*, **22**, 45–55.

Lozano, F. (2000). Model selection using Rademacher penalization. In *Proc. 2nd ICSC Symp. on Neural Comp.*

Lugosi, G. and Nobel, A. (1999). Adaptive model selection using empirical complexities. *Annals of Statistics*, **27**, 1830–1864.

Magdon-Ismail, M. and Mertsalov, K. (2010). A permutation approach to validation. In *Proc. 10th SIAM International Conference on Data Mining (SDM)*.

Massart, P. (2000). Some applications of concentration inequalities to statistics. *Annales de la Faculté des Sciencies de Toulouse*, **X**, 245–303.

McDiarmid, C. (1989). On the method of bounded differences. In *Surveys in Combinatorics*, pages 148–188. Cambridge University Press.

Mosier, C. I. (1951). Symposium: The need and means of cross-validation: I problem and designs of cross validation. *Education and Psychology Measurement*, **11**, 5–11.

Shawe-Taylor, J. and Cristianini, N. (2004). *Kernel Methods for Pattern Analysis*. Camb. Univ. Press.

Shawe-Taylor, J., Bartlett, P. L., Williamson, R. C., and Anthony, M. (1998). Structural risk minimization over data dependent hierarchies. *IEEE Transactions on Information Theory*, **44**, 1926–1940.

Stone, M. (1974). Cross validatory choice and assessment of statistical predictions. *Journal of the Royal Statistical Society*, **36**(2), 111–147.

Vapnik, V. N. and Chervonenkis, A. (1971). On the uniform convergence of relative frequencies of events to their pr obabilities. *Theory of Probability and its Applications*, **16**, 264–280.

Vapnik, V. N., Levin, E., and Le Cun, Y. (1994). Measuring the VC-dimension of a learning machine. *Neural Computation*, **6**(5), 851–876.

Wang, J. and Shen, X. (2006). Estimation of generalization error: random and fixed inputs. *Statistica Sinica*, **16**, 569–588.

Wherry, R. J. (1931). A new formula for predicting the shrinkage of the multiple correlation coefficient. *Annals of Mathematical Statistics*, **2**, 440–457.

Wherry, R. J. (1951). Symposium: The need and means of cross-validation: III comparison of cross validation with statistical inference of betas and multiple r from a single sample. *Education and Psychology Measurement*, **11**, 23–28.

Wiklund, S., Nilsson, D., Eriksson, L., Sjostrom, M., Wold, S., and Faber, K. (2007). A randomization test for PLS component selection. *Journal of Chemometrics*, **21**(10-11), 427–439.

